# The Effect of Correlations on the Fisher Information of Population Codes

**Hyoungsoo Yoon**
hyoung@fiz.huji.ac.il

**Haim Sompolinsky**
haim@fiz.huji.ac.il

Racah Institute of Physics and Center for Neural Computation
Hebrew University, Jerusalem 91904, Israel

## Abstract

We study the effect of correlated noise on the accuracy of population coding using a model of a population of neurons that are broadly tuned to an angle in two-dimension. The fluctuations in the neuronal activity is modeled as a Gaussian noise with pairwise correlations which decays exponentially with the difference between the preferred orientations of the pair. By calculating the Fisher information of the system, we show that in the biologically relevant regime of parameters positive correlations decrease the estimation capability of the network relative to the uncorrelated population. Moreover strong positive correlations result in information capacity which saturates to a finite value as the number of cells in the population grows. In contrast, negative correlations substantially increase the information capacity of the neuronal population.

## 1  Introduction

In many neural systems, information regarding sensory inputs or (intended) motor outputs is found to be distributed throughout a localized pool of neurons. It is generally believed that one of the main characteristics of the population coding scheme is its redundancy in representing information (Paradiso 1988; Snippe and Koenderink 1992a; Seung and Sompolinsky 1993). Hence the intrinsic neuronal noise, which has detrimental impact on the information processing capability, is expected to be compensated by increasing the number of neurons in a pool. Although this expectation is universally true for an ensemble of neurons whose stochastic variabilities are statistically independent, a general theory of the efficiency of population coding when the neuronal noise is correlated within the population, has been lacking. The conventional wisdom has been that the correlated variability limits the information processing capacity of neuronal ensembles (Zohary, Shadlen, and Newsome 1994).

However, detailed studies of simple models of a correlated population that code for a single real-valued parameter led to apparently contradicting claims. Snippe and Koenderink (Snippe and Koenderink 1992b) conclude that depending on the details of the correlations, such as their spatial range, they may either increase or decrease the information capacity relative to the uncorrelated one. Recently, Abbott and Dayan (Abbott and Dayan 1998) claimed that in many cases correlated noise improves the accuracy of population code. Furthermore, even when the information is decreased it still grows linearly with the size of the population. If true, this conclusion has an important implication on the utility of using a large population to improve the estimation accuracy. Since cross-correlations in neuronal activity are frequently observed in both primary sensory and motor areas (Fetz, Yoyama, and Smith 1991; Lee, Port, Kruse, and Georgopoulos 1998), understanding the effect of noise correlation in biologically relevant situations is of great importance.

In this paper we present an analytical study of the effect of noise correlations on the population coding of a pool of cells that code for a single one-dimensional variable, an angle on a plane, *e.g.*, an orientation of a visual stimulus, or the direction of an arm movement. By assuming that the noise follows the multivariate Gaussian distribution, we investigate analytically the effect of correlation on the Fisher information. This model is similar to that considered in (Snippe and Koenderink 1992b; Abbott and Dayan 1998). By analyzing its behavior in the biologically relevant regime of tuning width and correlation range, we derive general conclusions about the effect of the correlations on the information capacity of the population.

## 2 Population Coding with Correlated Noise

We consider a population of $N$ neurons which respond to a stimulus characterized by an angle $\theta$, where $-\pi < \theta \leq \pi$. The activity of each neuron (indexed by $i$) is assumed to be Gaussian with a mean $f_i(\theta)$ which represents its tuning curve, and a uniform variance $a$. The noise is assumed to be pairwise-correlated throughout the population. Hence the activity profile of the whole population, $R = \{r_1, r_2, \cdots, r_N\}$, given a stimulus $\theta$, follows the following multivariate Gaussian distribution.

$$P(R|\theta) = \mathcal{N} \exp \left( -\frac{1}{2} \sum_{i,j} (r_i - f_i(\theta)) C_{ij}^{-1} (r_j - f_j(\theta)) \right) \qquad (1)$$

where $\mathcal{N}$ is a normalization constant and $C_{ij}$ is the correlation matrix.

$$C_{ij} = a\delta_{ij} + b_{ij}(1 - \delta_{ij}). \qquad (2)$$

It is assumed that the tuning curves of all the neurons are identical in form but peaked at different angles, that is $f_i(\theta) = f(\theta - \phi_i)$ where the preferred angles $\phi_i$ are distributed *uniformly* from $-\pi$ to $\pi$ with a lattice spacing, $\omega$, which is equal to $2\pi/N$. We further assume that the noise correlation between a pair of neurons is only a function of their preferred angle difference, i.e., $b_{ij}(\theta) = b(\|\phi_i - \phi_j\|)$ where $\|\theta_1 - \theta_2\|$ is defined to be the relative angle between $\theta_1$ and $\theta_2$, and hence its maximum value is $\pi$. A decrease in the magnitude of neuronal correlations with the dissimilarity in the preferred stimulus is often observed in cortical areas. We model this by exponentially decaying correlations

$$b_{ij} = b \exp(-\frac{\|\phi_i - \phi_j\|}{\rho}) \qquad (3)$$

where $\rho$ specifies the angular correlation length.

The amount of information that can be extracted from the above population will depend on the decoding scheme. A convenient measure of the information capacity in the population is given by the Fisher information, which in our case is (for a given stimulus $\theta$)

$$J(\theta) = \sum_{i,j} g_i C_{ij}^{-1} g_j \tag{4}$$

where

$$g_i(\theta) \equiv \frac{\partial f_i(\theta)}{\partial \theta}. \tag{5}$$

The utility of this measure follows from the well known Cramér-Rao bound for the variance of any unbiased estimators, *i.e.*, $\langle (\theta - \hat{\theta})^2 \rangle \geq 1/J(\theta)$. For the rest of this paper, we will concentrate on the Fisher information as a function of the noise correlation parameters, $b$ and $\rho$, as well as the population size $N$.

## 3 Results

In the case of uncorrelated population ($b = 0$), the Fisher information is given by (Seung and Sompolinsky 1993)

$$J_o = \frac{N}{a} \sum_n |g_n|^2 \tag{6}$$

where $g_n$ is the Fourier transform of $g_j$, defined by

$$g_n \equiv \frac{1}{N} \sum_j e^{-in\phi_j} g_j. \tag{7}$$

The mode number $n$ is an integer running from $-\frac{N-1}{2}$ to $\frac{N-1}{2}$ (for odd $N$) and $\phi_i = -\pi(N+1)/N + i\omega$, $i = 1, \dots, N$. Likewise, in the case of $b \neq 0$, $J$ is given by

$$J = N \sum_n \frac{|g_n|^2}{C_n} \tag{8}$$

where $C_n$ are the eigenvalues of the covariance matrix,

$$
\begin{aligned}
C_n &\equiv \frac{1}{N} \sum_{i,j} e^{-in(\phi_i - \phi_j)} C_{ij} \\
&= (a - 2b) + 2b \frac{1 - \lambda \cos(n\omega) - (-1)^n \lambda^{\frac{N+1}{2}} \cos(n\omega)(1 - \lambda)}{1 - 2\lambda \cos(n\omega) + \lambda^2}
\end{aligned} \tag{9}
$$

where $\omega = \frac{2\pi}{N}$, $\lambda = e^{-\omega/\rho}$, and $N$ is assumed to be an odd integer. Note that the covariance matrix $C_{ij}$ remains positive definite as long as

$$-\frac{1}{2N} \frac{1 - \lambda}{\lambda(1 - \lambda^{\frac{N-1}{2}})} < \frac{b}{a} < 1 \tag{10}$$

where the lower bound holds for general $N$ while the upper bound is valid for large $N$.

To evaluate the effect of correlations in a large population it is important to specify the appropriate scales of the system parameters. We consider here the biologically relevant case of broadly tuned neurons that have a smoothly varying tuning curve with a single peak. When the tuning curve is smoothly varying, $|g_n|^2$ will be a rapidly decaying function as $n$ increases beyond a characteristic value which is

proportional to the inverse of the tuning width, $\sigma$. We further assume a broad tuning, namely that the tuning curve spans a substantial fraction of the angular extent. This is consistent with the observed typical values of half-width at half height in visual and motor areas, which range from 20 to 60 degrees. Likewise, it is reasonable to assume that the angular correlation length $\rho$ spans a substantial fraction of the entire angular range. This broad tuning of correlations with respect to the difference in the preferred angles is commonly observed in cortex (Fetz, Yoyama, and Smith 1991; Lee, Port, Kruse, and Georgopoulos 1998). To capture these features we will consider the limit of large $N$ while keeping the parameters $\rho$ and $\sigma$ constant. Note that keeping $\sigma$ of order 1 implies that substantial contributions to Eq. (8) come only from $n$ which remain of order 1 as $N$ increases. On the other hand, given the enormous variability in the strength of the observed cross-correlations between pairs of neurons in cortex, we do not restrict the value of $b$ at this point.

Incorporating the above scaling we find that when $N$ is large $J$ is given by

$$J = \frac{N}{a} \sum_n |g_n|^2 \frac{\rho^{-2} + n^2}{\rho^{-2} + n^2 + (\frac{2b}{a\omega\rho})(1 - (-1)^n e^{-\pi/\rho})}. \tag{11}$$

Inspection of the denominator in the above equation clearly shows that for all positive values of $b$, $J$ is smaller than $J_o$. On the other hand, when $b$ is negative $J$ is larger than $J_o$. To estimate the magnitude of these effects we consider below three different regimes.

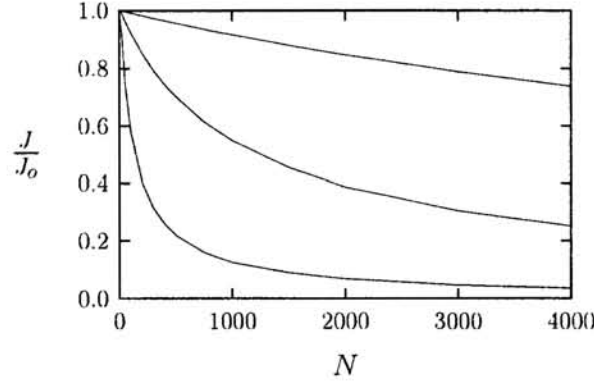

Figure 1: Normalized Fisher information when $\rho \sim \mathcal{O}(1)$ ($\rho = 0.25\pi$ was used). $a = 1$ and $b = 0.1, 0.01,$ and $0.001$ from the bottom. We used a circular Gaussian tuning curve, Eq. (13), with $f_{\max} = 10$ and $\sigma = 0.2\pi$.

**Strong positive correlations:** We first discuss the regime of strong positive correlations, by which we mean that $0 < b/a \sim \mathcal{O}(1)$. In this case the second term in the denominator of Eq. (11) is of order $N$ and Eq. (11) becomes

$$J = \frac{\pi\rho}{b} \sum_n |g_n|^2 \frac{\rho^{-2} + n^2}{1 - (-1)^n e^{-\pi/\rho}}. \tag{12}$$

This result implies that in this regime the Fisher information in the entire population does not scale linearly with the population size $N$ but saturates to a size-independent finite limit. Thus, for these strong correlations, although the number of neurons in the population may be large, the number of independent degrees of freedom is small.

We demonstrate the above phenomenon by a numerical evaluation of $J$ for the following choice of tuning curve

$$f(\theta) = f_{\max} \exp\left((\cos(\theta) - 1)/\sigma^2\right) \tag{13}$$

with $\sigma = 0.2\pi$. The results are shown in Fig. 1 and Fig. 2. The results of Fig. 1 clearly show the substantial decrease in $J$ as $b$ increases. The reduction in $J/J_o$ when $b \sim \mathcal{O}(1)$ indicates that $J$ does not scale with $N$ in this limit. Fig. 2 shows the saturation of $J$ when $N$ increases. For $\rho = 0.1$ and 1 ((c) and (d)), $J$ saturates at about $N = 100$, which means that for these parameter values the network contains at most 100 independent degrees of freedom. When the correlation range becomes either smaller or bigger, the saturation becomes less prominent ((a) and (b)), which is further explained later in the text.

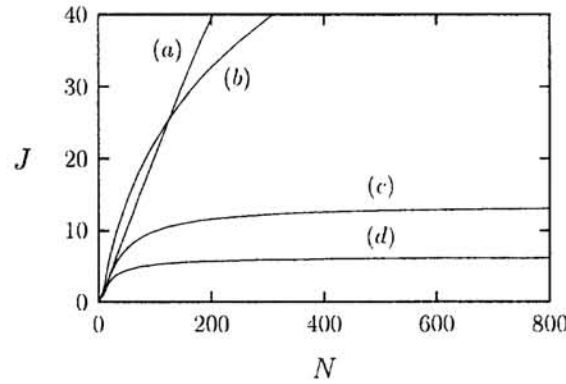

Figure 2: Saturation of Fisher information with the correlation coefficient kept fixed; $a = 1$ and $b = 0.5$. Both $\rho \sim \mathcal{O}(1)$ ((c) $\rho = 0.1$ and (d) $\rho = 1$) and other extreme limits ((a) $\rho = 0.01$ and (b) $\rho = 10$) are shown. Tuning curve with $f_{\mathrm{max}} = 1$ and $\sigma = 0.2\pi$ was used for all four curves.

**Weak positive correlations:** This regime is defined formally by positive values of $b$ which scale as $b/a \sim \mathcal{O}(\frac{1}{N})$. In this case, while $J$ is still smaller than $J_o$ the suppressive effects of the correlations are not as strong as in the first case. This is shown in Fig. 3 (bottom traces) for $N = 1000$. While $J$ is less than $J_o$, it is still a substantial fraction of $J_o$, indicating $J$ is of order $N$.

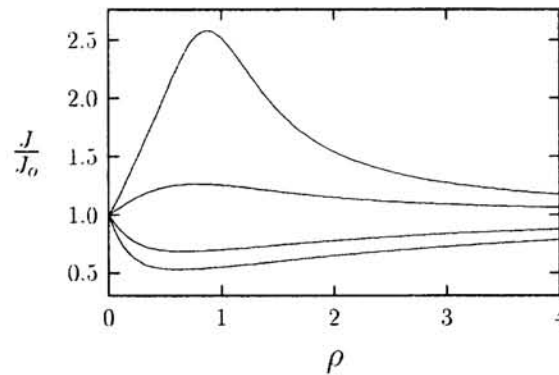

Figure 3: Normalized Fisher information when $\rho \sim \mathcal{O}(1)$ and $b/a \sim \mathcal{O}(\frac{1}{N})$. $N = 1000$, $a = 1$, $f_{\mathrm{max}} = 10$, and $\sigma = 0.2\pi$. The top curves represent negative $b$ ($b = -0.005$ and $-0.002$ from the top) and the bottom ones positive $b$ ($b = 0.01$ and 0.005 from the bottom).

**Weak negative correlations:** So far we have considered the case of positive $b$. As stated above, Eq. (11) implies that when $b < 0$, $J > J_o$. The lower bound of $b$ (Eq. (10)) means that when the correlations are negative and $\rho$ is of order 1 the amplitude of the correlations must be small. It scales as $b/a = \hat{b}/N$ with $\hat{b}$ which is of order 1 and is larger than $\hat{b}_{\mathrm{min}} = -(\pi/\rho)/(1 - \exp(-\pi/\rho))$. In this regime $(J - J_o)/N$ retains a finite positive value even for large $N$. This enhancement can

be made large if $\hat{b}$ comes close to $\hat{b}_{\min}$. This behavior is shown in Fig. 3 (upper traces). Note that, for both positive and negative weak correlations, the curves have peaks around a characteristic length scale $\rho \sim \sigma$, which is $0.2\pi$ in this figure.

**Extremely long and short range correlations:** Calculation with strictly uniform correlations, *i.e.*, $b_{ij} = b$, shows that in this case the positive correlations enhance the Fisher information of the system, leading to claims that this might be a generic result (Abbott and Dayan 1998). Here we show that this behavior is special to cases where the correlations essentially do not vary in strength. We consider the case $\rho \sim \mathcal{O}(N)$. This means that the strength of the correlations is the same for all the neurons up to a correction of order $1/N$. In this limit Eq. (11) is not valid, and the Fisher information is obtained from Eq. (8) and Eq. (9),

$$J = \frac{N}{a-b} \sum_{\text{even}} |g_n|^2 + N \sum_{\text{odd}} \frac{|g_n|^2}{a - b + b/(n^2 \varrho)} \tag{14}$$

where $\varrho = \omega\rho/4$. Note that even in this extreme regime, only for $\varrho > 1$ is $J$ guaranteed to be always larger than $J_o$. Below this value the sign of $J - J_o$ depends on the particular shape of the tuning curve and the value of $b$. In fact, a more detailed analysis (Yoon and Sompolinsky 1998) shows that as soon as $\rho \ll \mathcal{O}(\sqrt{N})$, $J - J_o < 0$, as in the case of $\rho \sim \mathcal{O}(1)$ discussed above. The crossover between these two opposite behaviors is shown in Fig. 4. For comparison the case with $\rho \sim \mathcal{O}(1)$ is also shown.

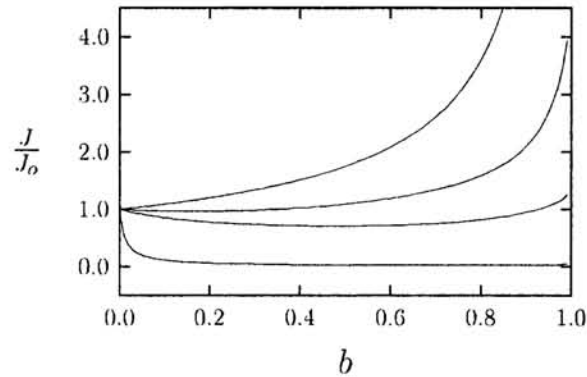

Figure 4: Normalized Fisher information when $b/a \sim \mathcal{O}(1)$. $N = 1000$ and $a = 1$. When $\rho \sim \mathcal{O}(1)$, increasing $b$ always decreases the Fisher information (bottom curve $\rho = 0.25\pi$). However, this trend is reversed when $\rho \sim \mathcal{O}(\sqrt{N})$ and when $\rho > \frac{2}{\pi}N$ $J - J_o$ becomes always positive. From the top $\rho = 400, 50,$ and $25$.

Another extreme regime is where the correlation length $\rho$ scales as $1/N$ but the tuning width remains of order 1. This means that a given neuron is correlated with a small number of its immediate neighbors, which remains finite as $N \to \infty$. In this limit, the Fisher information becomes, again from Eq. (8) and Eq. (9),

$$J = \frac{N(\lambda^{-1} - 1)}{a(\lambda^{-1} - 1) + 2b} \sum_{n} |g_n|^2. \tag{15}$$

In this case, the behavior of $J$ is similar to the cases of weak correlations discussed above. The information remains of order $N$ but the sign of $J - J_o$ depends on the sign of $b$. Thus, when the amplitude of the positive correlation function is $\mathcal{O}(1)$, $J$ increases linearly with $N$ in the two opposite extremes of very large and very small $\rho$ as shown in Fig. 2 ((a) and (b)).

# 4 Discussion

In this paper we have studied the effect of correlated variability of neuronal activity on the maximum accuracy of the population coding. We have shown that the effect of correlation on the information capacity of the population crucially depends on the scale of correlation length. We argue that for the sensory and motor areas which are presumed to utilize population coding, the tuning of both the correlations and the mean response profile is broad and of the same order. This implies that each neuron is correlated with a finite fraction of the total number of neurons, $N$, and a given stimulus activates a finite fraction of $N$. We show that in this regime positive correlations always decrease the information. When they are strong enough in amplitude they reduce the number of independent degrees of freedom to a finite number even for large population. Only in the extreme case of almost uniform correlations the information capacity is enhanced. This is reasonable since to overcome the positive correlations one needs to subtract the responses of different neurons. But in general this will reduce their signal by a larger amount. When the correlations are uniform, the reduction of the correlated noise by subtraction is perfect and can be made in a manner that will little affect the signal component.

## Acknowledgments

H.S. acknowledges helpful discussions with Larry Abbott and Sebastian Seung. This research is partially supported by the Fund for Basic Research of the Israeli Academy of Science and by a grant from the United States-Israel Binational Science Foundation (BSF), Jerusalem, Israel.

## References

L. F. Abbott and P. Dayan (1998). The effect of correlated variability on the accuracy of a population code. *Neural Comp.*, in press.

E. Fetz, K. Yoyama, and W. Smith (1991). Synaptic interactions between cortical neurons. In A. Peters and E. G. Jones (Eds.), *Cerebral Cortex*, Volume 9. New York: Plenum Press.

D. Lee, N. L. Port, W. Kruse, and A. P. Georgopoulos (1998). Variability and correlated noise in the discharge of neurons in motor and parietal areas of the primate cortex. *J. Neurosci. 18*, 1161–1170.

M. A. Paradiso (1988). A theory for the use of visual orientation information which exploits the columnar structure of striate cortex. *Biol. Cybern. 58*, 35–49.

H. S. Seung and H. Sompolinsky (1993). Simple models for reading neuronal population codes. *Proc. Natl. Acad. Sci. USA 90*, 10749–10753.

H. P. Snippe and J. J. Koenderink (1992a). Discrimination thresholds for channel-coded systems. *Biol. Cybern. 66*, 543–551.

H. P. Snippe and J. J. Koenderink (1992b). Information in channel-coded system: correlated receivers. *Biol. Cybern. 67*, 183–190.

H. Yoon and H. Sompolinsky (1998). Population coding in neuronal systems with correlated noise, preprint.

E. Zohary, M. N. Shadlen, and W. T. Newsome (1994). Correlated neuronal discharge rate and its implications for psychophysical performance. *Nature 370*, 140–143.